# Robust Fisher Discriminant Analysis

**Seung-Jean Kim    Alessandro Magnani    Stephen P. Boyd**
Information Systems Laboratory
Electrical Engineering Department, Stanford University
Stanford, CA 94305-9510
sjkim@stanford.edu    alem@stanford.edu    boyd@stanford.edu

## Abstract

Fisher linear discriminant analysis (LDA) can be sensitive to the problem data. Robust Fisher LDA can systematically alleviate the sensitivity problem by explicitly incorporating a model of data uncertainty in a classification problem and optimizing for the worst-case scenario under this model. The main contribution of this paper is show that with general convex uncertainty models on the problem data, robust Fisher LDA can be carried out using convex optimization. For a certain type of product form uncertainty model, robust Fisher LDA can be carried out at a cost comparable to standard Fisher LDA. The method is demonstrated with some numerical examples. Finally, we show how to extend these results to robust kernel Fisher discriminant analysis, *i.e.*, robust Fisher LDA in a high dimensional feature space.

## 1  Introduction

Fisher linear discriminant analysis (LDA), a widely-used technique for pattern classification, finds a linear discriminant that yields optimal discrimination between two classes which can be identified with two random variables, say $\mathbf{X}$ and $\mathbf{Y}$ in $\mathbb{R}^n$. For a (linear) discriminant characterized by $w \in \mathbb{R}^n$, the degree of discrimination is measured by the Fisher discriminant ratio

$$f(w, \mu_x, \mu_y, \Sigma_x, \Sigma_y) = \frac{w^T(\mu_x - \mu_y)(\mu_x - \mu_y)^T w}{w^T(\Sigma_x + \Sigma_y)w} = \frac{(w^T(\mu_x - \mu_y))^2}{w^T(\Sigma_x + \Sigma_y)w},$$

where $\mu_x$ and $\Sigma_x$ ($\mu_y$ and $\Sigma_y$) denote the mean and covariance of $\mathbf{X}$ ($\mathbf{Y}$). A discriminant that maximizes the Fisher discriminant ratio is given by

$$w^{\mathrm{nom}} = (\Sigma_x + \Sigma_y)^{-1}(\mu_x - \mu_y),$$

which gives the maximum Fisher discriminant ratio

$$(\mu_x - \mu_y)^T(\Sigma_x + \Sigma_y)^{-1}(\mu_x - \mu_y) = \max_{w \neq 0} f(w, \mu_x, \mu_y, \Sigma_x, \Sigma_y).$$

In applications, the problem data $\mu_x$, $\mu_y$, $\Sigma_x$, and $\Sigma_y$ are not known but are estimated from sample data. Fisher LDA can be sensitive to the problem data: the discriminant $w^{\mathrm{nom}}$ computed from an estimate of the parameters $\mu_x$, $\mu_y$, $\Sigma_x$, and $\Sigma_y$ can give very

poor discrimination for another set of problem data that is also a reasonable estimate of the parameters. In this paper, we attempt to systematically alleviate this sensitivity problem by explicitly incorporating a model of data uncertainty in the classification problem and optimizing for the worst-case scenario under this model.

We assume that the problem data $\mu_x$, $\mu_y$, $\Sigma_x$, and $\Sigma_y$ are uncertain, but known to belong to a convex compact subset $\mathcal{U}$ of $\mathbb{R}^n \times \mathbb{R}^n \times \mathbb{S}_{++}^n \times \mathbb{S}_{++}^n$. Here we use $\mathbb{S}_{++}^n$ ($\mathbb{S}_+^n$) to denote the set of all $n \times n$ symmetric positive definite (semidefinite) matrices. We make one technical assumption: for each $(\mu_x, \mu_y, \Sigma_x, \Sigma_y) \in \mathcal{U}$, we have $\mu_x \neq \mu_y$. This assumption simply means that for each possible value of the means and covariances, the classes are distinguishable via Fisher LDA.

The *worst-case analysis problem* of finding the worst-case means and covariances for a given discriminant $w$ can be written as

$$
\begin{array}{ll}
\text{minimize} & f(w, \mu_x, \mu_y, \Sigma_x, \Sigma_y) \\
\text{subject to} & (\mu_x, \mu_y, \Sigma_x, \Sigma_y) \in \mathcal{U},
\end{array}
\tag{1}
$$

with variables $\mu_x$, $\mu_y$, $\Sigma_x$, and $\Sigma_y$. The optimal value of this problem is the *worst-case Fisher discriminant ratio* (over the class $\mathcal{U}$ of possible means and covariances), and any optimal points for this problem are called *worst-case means and covariances*. These depend on $w$.

We will show in §2 that (1) is a convex optimization problem, since the Fisher discriminant ratio is a convex function of $\mu_x$, $\mu_y$, $\Sigma_x$, $\Sigma_y$ for a given discriminant $w$. As a result, it is computationally tractable to find the worst-case performance of a discriminant $w$ over the set of possible means and covariances.

The *robust Fisher LDA problem* is to find a discriminant that maximizes the worst-case Fisher discriminant ratio. This can be cast as the optimization problem

$$
\begin{array}{ll}
\text{maximize} & \min\limits_{(\mu_x, \mu_y, \Sigma_x, \Sigma_y) \in \mathcal{U}} f(w, \mu_x, \mu_y, \Sigma_x, \Sigma_y) \\
\text{subject to} & w \neq 0,
\end{array}
\tag{2}
$$

with variable $w$. We denote any optimal $w$ for this problem as $w^\star$. Here we choose a linear discriminant that maximizes the Fisher discrimination ratio, with the worst possible means and covariances that are consistent with our data uncertainty model.

The main result of this paper is to give an effective method for solving the robust Fisher LDA problem (2). We will show in §2 that the robust optimal Fisher discriminant $w^\star$ can be found as follows. First, we solve the (convex) optimization problem

$$
\begin{array}{ll}
\text{minimize} & \max\limits_{w \neq 0} f(w, \mu_x, \mu_y, \Sigma_x, \Sigma_y) = (\mu_x - \mu_y)^T (\Sigma_x + \Sigma_y)^{-1} (\mu_x - \mu_y) \\
\text{subject to} & (\mu_x, \mu_y, \Sigma_x, \Sigma_y) \in \mathcal{U},
\end{array}
\tag{3}
$$

with variables $(\mu_x, \mu_y, \Sigma_x, \Sigma_y)$. Let $(\mu_x^\star, \mu_y^\star, \Sigma_x^\star, \Sigma_y^\star)$ denote any optimal point. Then the discriminant

$$
w^\star = \left( \Sigma_x^\star + \Sigma_y^\star \right)^{-1} \left( \mu_x^\star - \mu_y^\star \right)
\tag{4}
$$

is a robust optimal Fisher discriminant, *i.e.*, it is optimal for (2). Moreover, we will see that $\mu_x^\star$, $\mu_y^\star$ and $\Sigma_x^\star$, $\Sigma_y^\star$ are worst-case means and covariances for the robust optimal Fisher discriminant $w^\star$. Since convex optimization problems are tractable, this means that we have a *tractable general method* for computing a robust optimal Fisher discriminant.

A robust Fisher discriminant problem of modest size can be solved by standard convex optimization methods, *e.g.*, interior-point methods [3]. For some special forms of the uncertainty model, the robust optimal Fisher discriminant can be solved more efficiently than by a general convex optimization formulation. In §3, we consider an important special form for $\mathcal{U}$ for which a more efficient formulation can be given.

In comparison with the 'nominal' Fisher LDA, which is based on the means and covariances estimated from the sample data set without considering the estimation error, the robust Fisher LDA performs well even when the sample size used to estimate the means and covariances is small, resulting in estimates which are not accurate. This will be demonstrated with some numerical examples in §4.

Recently, there has been a growing interest in kernel Fisher discriminant analysis *i.e.*, Fisher LDA in a higher dimensional feature space, *e.g.*, [7]. Our results can be extended to robust kernel Fisher discriminant analysis under certain uncertainty models. This will be briefly discussed in §5.

Various types of robust classification problems have been considered in the prior literature, *e.g.*, [2, 5, 6]. Most of the research has focused on formulating robust classification problems that can be efficiently solved via convex optimization. In particular, the robust classification method developed in [6] is based on the criterion

$$g(w, \mu_x, \mu_y, \Sigma_x, \Sigma_y) = \frac{|w^T(\mu_x - \mu_y)|}{(w^T\Sigma_x w)^{1/2} + (w^T\Sigma_y w)^{1/2}},$$

which is similar to the Fisher discriminant ratio $f$. With a specific uncertainty model on the means and covariances, the robust classification problem with discrimination criterion $g$ can be cast as a second-order cone program, a special type of convex optimization problem [5]. With general uncertainty models, however, it is not clear whether robust discriminant analysis with $g$ can be performed via convex optimization.

## 2  Robust Fisher LDA

We first consider the worst-case analysis problem (1). Here we consider the discriminant $w$ as fixed, and the parameters $\mu_x$, $\mu_y$, $\Sigma_x$, and $\Sigma_y$ are variables, constrained to lie in the convex uncertainty set $\mathcal{U}$. To show that (1) is a convex optimization problem, we must show that the Fisher discriminant ratio is a convex function of $\mu_x$, $\mu_y$, $\Sigma_x$, and $\Sigma_y$. To show this, we express the Fisher discriminant ratio $f$ as the composition

$$f(w, \mu_x, \mu_y, \Sigma_x, \Sigma_y) = g(H(\mu_x, \mu_y, \Sigma_x, \Sigma_y)),$$

where $g(u, t) = u^2/t$ and $H$ is the function

$$H(\mu_x, \mu_y, \Sigma_x, \Sigma_y) = (w^T(\mu_x - \mu_y), w^T(\Sigma_x + \Sigma_y)w).$$

The function $H$ is linear (as a mapping from $\mu_x$, $\mu_y$, $\Sigma_x$, and $\Sigma_y$ into $\mathbb{R}^2$), and the function $g$ is convex (provided $t > 0$, which holds here). Thus, the composition $f$ is a convex function of $\mu_x$, $\mu_y$, $\Sigma_x$, and $\Sigma_y$. (See [3].)

Now we turn to the main result of this paper. Consider a function of the form

$$R(w, a, B) = \frac{(w^T a)^2}{w^T B w}, \tag{5}$$

which is the Rayleigh quotient for the matrix pair $aa^T \in \mathbb{S}^n_+$ and $B \in \mathbb{S}^n_{++}$, evaluated at $w$. The robust Fisher LDA problem (2) is equivalent to a problem of the form

$$\begin{aligned} \text{maximize} \quad & \min_{(a,B)\in\mathcal{V}} R(w, a, B) \\ \text{subject to} \quad & w \neq 0, \end{aligned} \tag{6}$$

where

$$a = \mu_x - \mu_y, \quad B = \Sigma_x + \Sigma_y, \quad \mathcal{V} = \{(\mu_x - \mu_y, \Sigma_x + \Sigma_y) \mid (\mu_x, \mu_y, \Sigma_x, \Sigma_y) \in \mathcal{U}\}. \tag{7}$$

(This equivalence means that robust FLDA is a special type of robust matched filtering problem studied in the 1980s; see, *e.g.*, [8] for more on robust matched filtering.)

We will prove a 'nonconventional' minimax theorem for a Rayleigh quotient of the form (5), which will establish the main result described in §1. To do this, we consider a problem of the form

$$\begin{array}{ll} \text{minimize} & a^T B^{-1} a \\ \text{subject to} & (a, B) \in \mathcal{V}, \end{array} \tag{8}$$

with variables $a \in \mathbb{R}^n$, $B \in \mathbb{S}_{++}^n$, and $\mathcal{V}$ is a convex compact subset of $\mathbb{R}^n \times \mathbb{S}_{++}^n$ such that for each $(a, B) \in \mathcal{V}$, $a$ is not zero. The objective of this problem is a matrix fractional function and so is convex on $\mathbb{R}^n \times \mathbb{S}_{++}^n$; see [3, §3.1.7]. Our problem (3) is the same as (8), with (7). It follows that (3) is a convex optimization problem.

The following theorem states the minimax theorem for the function $R$. While $R$ is convex in $(a, B)$ for fixed $w$, it is *not* concave in $w$ for fixed $(a, B)$, so conventional convex-concave minimax theorems do not apply here.

**Theorem 1.** *Let $(a^\star, B^\star)$ be an optimal solution to the problem (8), and let $w^\star = B^{\star -1} a^\star$. Then $(w^\star, a^\star, B^\star)$ satisfies the minimax property*

$$R(w^\star, a^\star, B^\star) = \max_{w \neq 0} \min_{(a,B) \in \mathcal{V}} R(w, a, B) = \min_{(a,B) \in \mathcal{V}} \max_{w \neq 0} R(w, a, B), \tag{9}$$

*and the saddle point property*

$$R(w, a^\star, B^\star) \leq R(w^\star, a^\star, B^\star) \leq R(w^\star, a, B), \ \forall w \in \mathbb{R}^n \backslash \{0\}, \ \forall (a, B) \in \mathcal{V}. \tag{10}$$

*Proof.* It suffices to prove (10), since the saddle point property (10) implies the minimax property (9) [1, §2.6]. We start by observing that $R(w, a^\star, B^\star)$ is maximized over nonzero $w \neq 0$ by $w^\star = B^{\star -1} a^\star$ (by the Cauchy-Schwartz inequality). What remains is to show

$$\min_{(a,B) \in \mathcal{V}} R(w^\star, a, B) = R(w^\star, a^\star, B^\star). \tag{11}$$

Since $a^\star$ and $B^\star$ are optimal for the convex problem (8) (by definition), they must satisfy the optimality condition

$$\left\langle \nabla_a (a^T B^{-1} a) \big|_{(a^\star, B^\star)}, (a - a^\star) \right\rangle + \left\langle \nabla_B (a^T B^{-1} a) \big|_{(a^\star, B^\star)}, (B - B^\star) \right\rangle$$
$$\geq 0, \quad \forall (a, B) \in \mathcal{V}$$

(see [3, §4.2.3]). Using $\nabla_a (a^T B^{-1} a) = 2B^{-1} a$, $\nabla_B (a^T B^{-1} a) = -B^{-1} a a^T B^{-1}$, and $\langle X, Y \rangle = \mathbf{Tr}(XY)$ for $X, Y \in \mathbb{S}^n$, where $\mathbf{Tr}$ denotes trace, we can express the optimality condition as

$$2a^{\star T} B^{\star -1} (a - a^\star) - \mathbf{Tr} B^{\star -1} a^\star a^{\star T} B^{\star -1} (B - B^\star) \geq 0, \quad \forall (a, B) \in \mathcal{V},$$

or equivalently,

$$2w^{\star T} (a - a^\star) - w^{\star T} (B - B^\star) w^\star \geq 0, \quad \forall (a, B) \in \mathcal{V}. \tag{12}$$

Now we turn to the convex optimization problem

$$\begin{array}{ll} \text{minimize} & R(w^\star, a, B) \\ \text{subject to} & (a, B) \in \mathcal{V}, \end{array} \tag{13}$$

with variables $(a, B)$. We will show that $(a^\star, B^\star)$ is optimal for this problem, which will establish (11).

A pair $(\bar{a}, \bar{B})$ is optimal for (13) if and only if

$$\left\langle \nabla_a \left. \frac{(w^{\star T} a)^2}{w^{\star T} B w^\star} \right|_{(\bar{a}, \bar{B})} , (a - \bar{a}) \right\rangle + \left\langle \nabla_B \left. \frac{(w^{\star T} a)^2}{w^{\star T} B w^\star} \right|_{(\bar{a}, \bar{B})} , (B - \bar{B}) \right\rangle \geq 0, \quad \forall (a, B) \in \mathcal{V}.$$

Using

$$\nabla_a \frac{(w^{\star T} a)^2}{w^{\star T} B w^\star} = 2 \frac{a^T w^\star}{w^\star B w^\star} w^\star, \qquad \nabla_B \frac{(w^{\star T} a)^2}{w^{\star T} B w^\star} = -\frac{(a^T w^\star)^2}{(w^{\star T} B w^\star)^2} w^\star w^{\star T},$$

the optimality condition can be written as

$$2 \frac{\bar{a}^T w^\star}{w^{\star T} \bar{B} w^\star} w^{\star T} (a - \bar{a}) - \mathbf{Tr} \frac{(\bar{a}^T w^\star)^2}{(w^{\star T} \bar{B} w^\star)^2} w^\star w^{\star T} (B - \bar{B})$$

$$= \quad 2 \frac{\bar{a}^T w^\star}{w^{\star T} \bar{B} w^\star} w^{\star T} (a - \bar{a}) - \frac{(\bar{a}^T w^\star)^2}{(w^{\star T} \bar{B} w^\star)^2} w^{\star T} (B - \bar{B}) w^\star$$

$$\geq \quad 0, \quad \forall (a, B) \in \mathcal{V}.$$

Substituting $\bar{a} = a^\star$, $\bar{B} = B^\star$, and noting that $a^{\star T} w^\star / w^{\star T} B^\star w^\star = 1$, the optimality condition reduces to

$$2 w^{\star T} (a - a^\star) - w^{\star T} (B - B^\star) w^\star \geq 0, \quad \forall (a, B) \in \mathcal{V},$$

which is precisely (12). Thus, we have shown that $(a^\star, B^\star)$ is optimal for (13), which in turn establishes (11). $\qquad \square$

## 3 Robust Fisher LDA with product form uncertainty models

In this section, we focus on robust Fisher LDA with the product form uncertainty model

$$\mathcal{U} = \mathcal{M} \times \mathcal{S}, \tag{14}$$

where $\mathcal{M}$ is the set of possible means and $\mathcal{S}$ is the set of possible covariances. For this model, the worst-case Fisher discriminant ratio can be written as

$$\min_{(\mu_x, \mu_y, \Sigma_x, \Sigma_y) \in \mathcal{U}} f(a, \mu_x, \mu_y, \Sigma_x, \Sigma_y) = \min_{(\mu_x, \mu_y) \in \mathcal{M}} \frac{(w^T (\mu_x - \mu_y))^2}{\max_{(\Sigma_x, \Sigma_y) \in \mathcal{S}} w^T (\Sigma_x + \Sigma_y) w}.$$

If we can find an analytic expression for $\max_{(\Sigma_x, \Sigma_y) \in \mathcal{S}} w^T (\Sigma_x + \Sigma_y) w$ (as a function of $w$), we can simplify the robust Fisher LDA problem.

As a more specific example, we consider the case in which $\mathcal{S}$ is given by

$$\begin{aligned} \mathcal{S} &= \mathcal{S}_x \times \mathcal{S}_y, \\ \mathcal{S}_x &= \{ \Sigma_x \mid \Sigma_x \succeq 0, \|\Sigma_x - \bar{\Sigma}_x\|_F \leq \delta_x \}, \\ \mathcal{S}_y &= \{ \Sigma_y \mid \Sigma_y \succeq 0, \|\Sigma_y - \bar{\Sigma}_y\|_F \leq \delta_y \}, \end{aligned} \tag{15}$$

where $\delta_x, \delta_y$ are positive constants, $\bar{\Sigma}_x, \bar{\Sigma}_y \in \mathbb{S}_{++}^n$, and $\|A\|_F$ denotes the Frobenius norm of $A$, *i.e.*, $\|A\|_F = (\sum_{i,j=1}^n A_{ij}^2)^{1/2}$. For this case, we have

$$\max_{(\Sigma_x, \Sigma_y) \in \mathcal{S}} w^T (\Sigma_x + \Sigma_y) w = w^T (\bar{\Sigma}_x + \bar{\Sigma}_y + (\delta_x + \delta_y) I) w. \tag{16}$$

Here we have used the fact that for given $\bar{\Sigma} \in \mathbb{S}_{++}^n$, $\max_{\|\Sigma - \bar{\Sigma}\|_F \leq \delta} x^T \Sigma x = x^T (\bar{\Sigma} + \delta I) x$ (see, *e.g.*, [6]). The worst-case Fisher discriminant ratio can be expressed as

$$\min_{(\mu_x, \mu_y) \in \mathcal{M}} \frac{(w^T (\mu_x - \mu_y))^2}{w^T (\bar{\Sigma}_x + \bar{\Sigma}_y + (\delta_x + \delta_y) I) w}.$$

This is the same worst-case Fisher discriminant ratio obtained for a problem in which the covariances are certain, *i.e.*, fixed to be $\bar{\Sigma}_x + \delta_x I$ and $\bar{\Sigma}_y + \delta_y I$, and the means lie in the set $\mathcal{M}$. We conclude that a robust optimal Fisher discriminant with the uncertainty model (14) in which $\mathcal{S}$ has the form (15) can be found by solving a robust Fisher LDA problem with these fixed values for the covariances. From the general solution method described in §1, it is given by

$$w^\star = \left( \bar{\Sigma}_x + \bar{\Sigma}_y + (\delta_x + \delta_y)I \right)^{-1} (\mu_x^\star - \mu_y^\star),$$

where $\mu_x^\star$ and $\mu_y^\star$ solve the convex optimization problem

$$
\begin{array}{ll}
\text{minimize} & (\mu_x - \mu_y)^T \left( \bar{\Sigma}_x + \bar{\Sigma}_y + (\delta_x + \delta_y)I \right)^{-1} (\mu_x - \mu_y) \\
\text{subject to} & (\mu_x, \mu_y) \in \mathcal{M},
\end{array}
\tag{17}
$$

with variables $\mu_x$ and $\mu_y$.

The problem (17) is relatively simple: it involves minimizing a convex quadratic function over the set of possible $\mu_x$ and $\mu_y$. For example, if $\mathcal{M}$ is a product of two ellipsoids, (*e.g.*, $\mu_x$ and $\mu_y$ each lie in some confidence ellipsoid) the problem (17) is to minimize a convex quadratic subject to two convex quadratic constraints. Such a problem is readily solved in $O(n^3)$ flops, since the dual problem has two variables, and evaluating the dual function and its derivatives can be done in $O(n^3)$ flops [3]. Thus, the effort to solve the robust is the same order (*i.e.*, $n^3$) as solving the nominal Fisher LDA (but with a substantially larger constant).

## 4  Numerical results

To demonstrate robust Fisher LDA, we use the sonar and ionosphere benchmark problems from the UCI repository (`www.ics.uci.edu/~mlearn/MLRepository.html`). The two benchmark problems have 208 and 351 points, respectively, and the dimension of each data point is 60 and 34, respectively. Each data set is randomly partitioned into a training set and a test set. We use the training set to compute the optimal discriminant and then test its performance using the test set. A larger training set typically gives better test performance. We let $\alpha$ denote the size of the training set, as a fraction of the total number of data points. For example, $\alpha = 0.3$ means that $30\%$ of the data points are used for training, and $70\%$ are used to test the resulting discriminant. For various values of $\alpha$, we generate 100 random partitions of the data (for each of the two benchmark problems), and collect the results.

We use the following uncertainty models for the means $\mu_x, \mu_y$ and the covariances $\Sigma_x, \Sigma_y$:

$$
\begin{array}{ll}
(\mu_x - \bar{\mu}_x)^T P_x (\mu_x - \bar{\mu}_x) \leq 1, & \|\Sigma_x - \bar{\Sigma}_x\|_F \leq \rho_x, \\
(\mu_y - \bar{\mu}_y)^T P_y (\mu_y - \bar{\mu}_y) \leq 1, & \|\Sigma_y - \bar{\Sigma}_y\|_F \leq \rho_y,
\end{array}
$$

Here the vectors $\bar{\mu}_x, \bar{\mu}_y$ represent the nominal means and the matrices $\bar{\Sigma}_x, \bar{\Sigma}_y$ represent the nominal covariances, and the matrices $P_x, P_y$ and the constants $\rho_x$ and $\rho_y$ represent the confidence regions. The parameters are estimated through a resampling technique [4] as follows. For a given training set we create 100 new sets by resampling the original training set with a uniform distribution over all the data points. For each of these sets we estimate its mean and covariance and then take their average values as the nominal mean and covariance. We also evaluate the covariance $\Sigma_\mu$ of all the means obtained with the resampling. We then take $P_x = \Sigma_\mu^{-1}/n$ and $P_y = \Sigma_\mu^{-1}/n$. This choice corresponds to a $50\%$ confidence ellipsoid in the case of a Gaussian distribution. The parameters $\rho_x$ and $\rho_y$ are taken to be the maximum deviations between the covariances and the average covariances in the Frobenius norm sense, over the resampling of the training set.

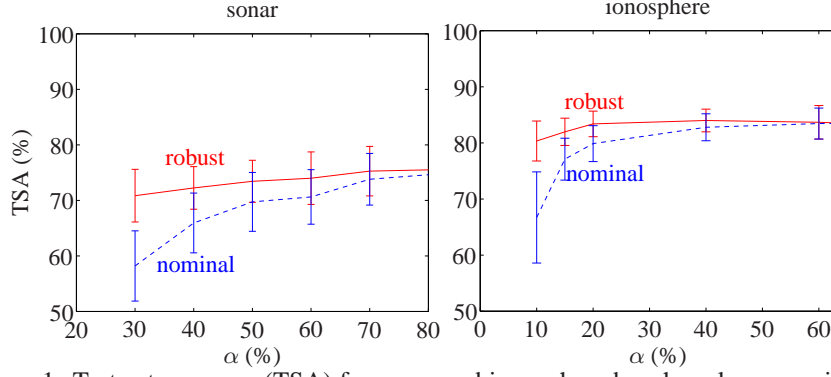

Figure 1: Test-set accuracy (TSA) for sonar and ionosphere benchmark versus size of the training set. The solid line represents the robust Fisher LDA results and the dotted line the nominal Fisher LDA results. The vertical bars represent the standard deviation.

Figure 1 summarizes the classification results. For each of our two problems, and for each value of $\alpha$, we show the average test set accuracy (TSA), as well as the standard deviation (over the $100$ instances of each problem with the given value of $\alpha$). The plots show the robust Fisher LDA performs substantially better than the nominal Fisher LDA for small training sets, but this performance gap disappears as the training set becomes larger.

## 5  Robust kernel Fisher discriminant analysis

In this section we show how to 'kernelize' the robust Fisher LDA. We will consider only a specific class of uncertainty models; the arguments we develop here can be extended to more general cases. In the kernel approach we map the problem to an higher dimensional space $\mathbb{R}^f$ via a mapping $\phi : \mathbb{R}^n \to \mathbb{R}^f$ so that the new decision boundary is more general and possibly nonlinear. Let the data be mapped as

$$x \to \phi(x) \sim (\bar{\mu}_{\phi(x)}, \bar{\Sigma}_{\phi(x)}), \quad y \to \phi(y) \sim (\bar{\mu}_{\phi(y)}, \bar{\Sigma}_{\phi(y)}).$$

The uncertainty model we consider has the form

$$
\begin{aligned}
\mu_{\phi(x)} - \mu_{\phi(y)} &= \bar{\mu}_{\phi(x)} - \bar{\mu}_{\phi(y)} + Pu_f, \quad \|u_f\| \leq 1, \\
\|\Sigma_{\phi(x)} - \bar{\Sigma}_{\phi(x)}\|_F &\leq \rho_x, \quad \|\Sigma_{\phi(y)} - \bar{\Sigma}_{\phi(y)}\|_F \leq \rho_y.
\end{aligned}
\tag{18}
$$

Here the vectors $\bar{\mu}_{\phi(x)}, \bar{\mu}_{\phi(y)}$ represent the nominal means, the matrices $\bar{\Sigma}_{\phi(x)}, \bar{\Sigma}_{\phi(y)}$ represent the nominal covariances, and the (positive semidefinite) matrix $P$ and the constants $\rho_x$ and $\rho_y$ represent the confidence regions in the feature space. The worst-case Fisher discriminant ratio in the feature space is then given by

$$
\min_{\|u_f\| \leq 1, \|\Sigma_{\phi(x)} - \bar{\Sigma}_{\phi(x)}\|_F \leq \rho_x, \|\Sigma_{\phi(y)} - \bar{\Sigma}_{\phi(y)}\|_F \leq \rho_y} \frac{(w_f^T(\bar{\mu}_{\phi(x)} - \bar{\mu}_{\phi(y)} + Pu_f))^2}{w_f^T(\Sigma_{\phi(x)} + \Sigma_{\phi(y)})w_f}.
$$

The robust kernel Fisher discriminant analysis problem is to find the discriminant in the feature space that maximizes this ratio.

Using the technique described in §3, we can see that the robust kernel Fisher discriminant analysis problem can be cast as

$$
\begin{aligned}
\text{maximize} \quad & \min_{\|u_f\| \leq 1} \frac{(w_f^T(\bar{\mu}_{\phi(x)} - \bar{\mu}_{\phi(y)} + Pu_f))^2}{w_f^T(\bar{\Sigma}_{\phi(x)} + \bar{\Sigma}_{\phi(y)} + (\rho_x + \rho_y)I)w_f} \\
\text{subject to} \quad & w_f \neq 0,
\end{aligned}
\tag{19}
$$

where the discriminant $w_f \in \mathbb{R}^f$ is defined in the new feature space.

To apply the kernel trick to the problem (19), the nonlinear decision boundary should be entirely expressed in terms of inner products of the mapped data only. The following proposition tells us a set of conditions to do so.

**Proposition 1.** *Given the sample points $\{x_i\}_{i=1}^{N_x}$ and $\{y_i\}_{i=1}^{N_y}$, suppose that $\bar{\mu}_{\phi(x)}, \bar{\mu}_{\phi(y)}$, $\bar{\Sigma}_{\phi(x)}, \bar{\Sigma}_{\phi(y)}$, and $P$ can be written as*

$$\bar{\mu}_{\phi(x)} = \sum_{i=1}^{N_x} \lambda_i \phi(x_i), \quad \bar{\mu}_{\phi(y)} = \sum_{i=1}^{N_y} \lambda_{i+N_x} \phi(y_i), \quad P = U \Upsilon U^T,$$

$$\bar{\Sigma}_{\phi(x)} = \sum_{i=1}^{N_x} \Lambda_{i,i} (\phi(x_i) - \bar{\mu}_{\phi(x)})(\phi(x_i) - \bar{\mu}_{\phi(x)})^T,$$

$$\bar{\Sigma}_{\phi(y)} = \sum_{i=1}^{N_y} \Lambda_{i+N_x, i+N_x} (\phi(y_i) - \bar{\mu}_{\phi(y)})(\phi(y_i) - \bar{\mu}_{\phi(y)})^T,$$

*where $\lambda \in \mathbb{R}^{N_x+N_y}$, $\Upsilon \in \mathbb{S}_+^{N_x+N_y}$, $\Lambda \in \mathbb{S}_+^{N_x+N_y}$ is a diagonal matrix, and $U$ is a matrix whose columns are the vectors $\{\phi(x_i) - \bar{\mu}_{\phi(x)}\}_{i=1}^{N_x}$ and $\{\phi(y_i) - \bar{\mu}_{\phi(y)}\}_{i=1}^{N_y}$. Denote as $\Phi$ the matrix whose columns are the vectors $\{\phi(x_i)\}_{i=1}^{N_x}$, $\{\phi(y_i)\}_{i=1}^{N_y}$ and define*

$$D_1 = K\beta, \quad D_2 = K(I - \lambda 1_N^T)\Upsilon(I - \lambda 1_N^T)K^T,$$

$$D_3 = K(I - \lambda 1_N^T)\Lambda(I - \lambda 1_N^T)K^T + (\rho_x + \rho_y)K, \quad D_4 = K,$$

*where $K$ is the kernel matrix $K_{ij} = (\Phi^T \Phi)_{ij}$, $1_N$ is a vector of ones of length $N_x + N_y$, and $\beta \in \mathbb{R}^{N_x+N_y}$ is such that $\beta_i = \lambda_i$ for $i = 1, \ldots, N_x$ and $\beta_i = -\lambda_i$ for $i = N_x + 1, \ldots, N_x + N_y$. Let $\nu^\star$ be an optimal solution of the problem*

$$\begin{aligned} \text{maximize} \quad & \min_{\xi^T D_4 \xi \le 1} \frac{\nu^T (D_1 + D_2\xi)(D_1 + D_2\xi)^T \nu}{\nu^T D_3 \nu} \\ \text{subject to} \quad & \nu \neq 0. \end{aligned} \tag{20}$$

*Then, $w_f^\star = \Phi \nu^\star$ is an optimal solution of the problem (19). Moreover, for every point $z \in \mathbb{R}^n$,*

$$w_f^{\star T} \phi(z) = \sum_{i=1}^{N_x} \nu_i^\star K(z, x_i) + \sum_{i=1}^{N_y} \nu_{i+N_x}^\star K(z, y_i). \tag{21}$$

Along the lines of the proofs of Corollary 5 in [6], we can prove this proposition.

## References

[1] D. Bertsekas, A. Nedić, and A. Ozdaglar. *Convex Analysis and Optimization*. Athena Scientific, 2003.

[2] C. Bhattacharyya. Second order cone programming formulations for feature selection. *Journal of Machine Learning Research*, 5:1417–1433, 2004.

[3] S. Boyd and L. Vandenberghe. *Convex Optimization*. Cambridge University Press, 2004.

[4] B. Efron and R.J. Tibshirani. *An Introduction to Bootstrap*. Chapman and Hall, London UK, 1993.

[5] K. Huang, H. Yang, I. King, M. Lyu, and L. Chan. The minimum error minimax probability machine. *Journal of Machine Learning Research*, 5:1253–1286, 2004.

[6] G. Lanckriet, L. El Ghaoui, C. Bhattacharyya, and M. Jordan. A robust minimax approach to classification. *Journal of Machine Learning Research*, 3:555–582, 2002.

[7] S. Mika, G. Rätsch, and K. Müller. A mathematical programming approach to the kernel Fisher algorithm, 2001. In *Advances in Neural Information Processing Systems*, 13, pp. 591-597, MIT Press.

[8] S. Verdú and H. Poor. On minimax robustness: A general approach and applications. *IEEE Transactions on Information Theory*, 30(2):328–340, 1984.
